# Generalized Belief Propagation

**Jonathan S. Yedidia**
MERL
201 Broadway
Cambridge, MA 02139
Phone: 617-621-7544
yedidia@merl.com

**William T. Freeman**
MERL
201 Broadway
Cambridge, MA 02139
Phone: 617-621-7527
freeman@merl.com

**Yair Weiss**
Computer Science Division
UC Berkeley, 485 Soda Hall
Berkeley, CA 94720-1776
Phone: 510-642-5029
yweiss@cs.berkeley.edu

## Abstract

Belief propagation (BP) was only supposed to work for tree-like networks but works surprisingly well in many applications involving networks with loops, including turbo codes. However, there has been little understanding of the algorithm or the nature of the solutions it finds for general graphs.

We show that BP can only converge to a stationary point of an approximate free energy, known as the Bethe free energy in statistical physics. This result characterizes BP fixed-points and makes connections with variational approaches to approximate inference.

More importantly, our analysis lets us build on the progress made in statistical physics since Bethe's approximation was introduced in 1935. Kikuchi and others have shown how to construct more accurate free energy approximations, of which Bethe's approximation is the simplest. Exploiting the insights from our analysis, we derive generalized belief propagation (GBP) versions of these Kikuchi approximations. These new message passing algorithms can be significantly more accurate than ordinary BP, at an adjustable increase in complexity. We illustrate such a new GBP algorithm on a grid Markov network and show that it gives much more accurate marginal probabilities than those found using ordinary BP.

## 1  Introduction

Local "belief propagation" (BP) algorithms such as those introduced by Pearl are guaranteed to converge to the correct marginal posterior probabilities in tree-like graphical models. For general networks with loops, the situation is much less clear. On the one hand, a number of researchers have empirically demonstrated good performance for BP algorithms applied to networks with loops. One dramatic case is the near Shannon-limit performance of "Turbo codes", whose decoding algorithm is equivalent to BP on a loopy network [2, 6]. For some problems in computer vision involving networks with loops, BP has also shown to be accurate and to converge very quickly [2, 1, 7]. On the other hand, for other networks with loops, BP may give poor results or fail to converge [7].

For a general graph, little has been understood about what approximation BP represents, and how it might be improved. This paper's goal is to provide that understanding and introduce a set of new algorithms resulting from that understanding. We show that BP is the first in a progression of local message-passing algorithms, each giving equivalent results to a corresponding approximation from statistical physics known as the "Kikuchi" approximation to the Gibbs free energy. These algorithms have the attractive property of being user-adjustable: by paying some additional computational cost, one can obtain considerable improvement in the accuracy of one's approximation, and can sometimes obtain a convergent message-passing algorithm when ordinary BP does not converge.

## 2 Belief propagation fixed-points are zero gradient points of the Bethe free energy

We assume that we are given an undirected graphical model of $N$ nodes with pair-wise potentials (a Markov network). Such a model is very general, as essentially any graphical model can be converted into this form. The state of each node $i$ is denoted by $x_i$, and the joint probability distribution function is given by

$$P(x_1, x_2, ..., x_N) = \frac{1}{Z} \prod_{ij} \psi_{ij}(x_i, x_j) \prod_i \psi_i(x_i) \tag{1}$$

where $\psi_i(x_i)$ is the local "evidence" for node $i$, $\psi_{ij}(x_i, x_j)$ is the compatibility matrix between nodes $i$ and $j$, and $Z$ is a normalization constant. Note that we are subsuming any fixed evidence nodes into our definition of $\psi_i(x_i)$.

The standard BP update rules are:

$$m_{ij}(x_j) \leftarrow \alpha \sum_{x_i} \psi_{ij}(x_i, x_j) \psi_i(x_i) \prod_{k \in N(i) \backslash j} m_{ki}(x_i) \tag{2}$$

$$b_i(x_i) \leftarrow \alpha \psi_i(x_i) \prod_{k \in N(i)} m_{ki}(x_i) \tag{3}$$

where $\alpha$ denotes a normalization constant and $N(i) \backslash j$ means all nodes neighboring node $i$, except $j$. Here $m_{ij}$ refers to the message that node $i$ sends to node $j$ and $b_i$ is the belief (approximate marginal posterior probability) at node $i$, obtained by multiplying all incoming messages to that node by the local evidence. Similarly, we can define the belief $b_{ij}(x_i, x_j)$ at the pair of nodes $(x_i, x_j)$ as the product of the local potentials and all messages incoming to the pair of nodes: $b_{ij}(x_i, x_j) = \alpha \phi_{ij}(x_i, x_j) \prod_{k \in N(i) \backslash j} m_{ki}(x_i) \prod_{l \in N(j) \backslash i} m_{lj}(x_j)$, where $\phi_{ij}(x_i, x_j) \equiv \psi_{ij}(x_i, x_j) \psi_i(x_i) \psi_j(x_j)$.

*Claim 1:* Let $\{m_{ij}\}$ be a set of BP messages and let $\{b_{ij}, b_i\}$ be the beliefs calculated from those messages. Then the beliefs are fixed-points of the BP algorithm if and only if they are zero gradient points of the Bethe free energy, $F_\beta$:

$$F_\beta(\{b_{ij}, b_i\}) = \sum_{ij} \sum_{x_i, x_j} b_{ij}(x_i, x_j) \left[ \ln b_{ij}(x_i, x_j) - \ln \phi_{ij}(x_i, x_j) \right]$$

$$- \sum_i (q_i - 1) \sum_{x_i} b_i(x_i) \left[ \ln b_i(x_i) - \ln \psi_i(x_i) \right] \tag{4}$$

subject to the normalization and marginalization constraints: $\sum_{x_i} b_i(x_i) = 1$, $\sum_{x_i} b_{ij}(x_i, x_j) = b_j(x_j)$. ($q_i$ is the number of neighbors of node $i$.)

To prove this claim we add Lagrange multipliers to form a Lagrangian $L$: $\lambda_{ij}(x_j)$ is the multiplier corresponding to the constraint that $b_{ij}(x_i, x_j)$ marginalizes down to $b_j(x_j)$, and $\gamma_{ij}, \gamma_i$ are multipliers corresponding to the normalization constraints. The equation $\frac{\partial L}{\partial b_{ij}(x_i,x_j)} = 0$ gives: $\ln b_{ij}(x_i, x_j) = \ln(\phi_{ij}(x_i, x_j)) + \lambda_{ij}(x_j) + \lambda_{ji}(x_i) + \gamma_{ij} - 1$. The equation $\frac{\partial L}{\partial b_i(x_i)} = 0$ gives: $(q_i - 1)(\ln b_i(x_i) + 1) = \ln \psi_i(x_i) + \sum_{j \in N(i)} \lambda_{ji}(x_i) + \gamma_i$. Setting $\lambda_{ij}(x_j) = \ln \prod_{k \in N(j) \backslash i} m_{kj}(x_j)$ and using the marginalization constraints, we find that the stationary conditions on the Lagrangian are equivalent to the BP fixed-point conditions. (Empirically, we find that *stable* BP fixed-points correspond to local *minima* of the Bethe free energy, rather than maxima or saddle-points.)

## 2.1 Implications

The fact that $F_\beta(\{b_{ij}, b_i\})$ is bounded below implies that the BP equations always possess a fixed-point (obtained at the global minimum of $F$). To our knowledge, this is the first proof of existence of fixed-points for a general graph with arbitrary potentials (see [9] for a complicated proof for a special case).

The free energy formulation clarifies the relationship to variational approaches which also minimize an approximate free energy [3]. For example, the mean field approximation finds a set of $\{b_i\}$ that minimize:

$$F_{MF}(\{b_i\}) = -\sum_{ij} \sum_{x_i, x_j} b_i(x_i) b_j(x_j) \ln \psi_{ij}(x_i, x_j) + \sum_i \sum_{x_i} b_i(x_i) \left[\ln b_i(x_i) - \ln \psi_i(x_i)\right]$$

(5)

subject to the constraint $\sum_i b_i(x_i) = 1$.

The BP free energy includes first-order terms $b_i(x_i)$ as well as second-order terms $b_{ij}(x_i, x_j)$, while the mean field free energy uses only the first order ones. It is easy to show that the BP free energy is exact for trees while the mean field one is not. Furthermore the optimization methods are different: typically $F_{MF}$ is minimized directly in the primal variables $\{b_i\}$ while $F_\beta$ is minimized using the messages, which are a combination of the dual variables $\{\lambda_{ij}(x_j)\}$.

Kabashima and Saad [4] have previously pointed out the correspondence between BP and the Bethe approximation (expressed using the TAP formalism) for some specific graphical models with random disorder. Our proof answers in the affirmative their question about whether there is a "deep general link between the two methods." [4]

## 3 Kikuchi Approximations to the Gibbs Free Energy

The Bethe approximation, for which the energy and entropy are approximated by terms that involve at most pairs of nodes, is the simplest version of the Kikuchi "cluster variational method." [5, 10] In a general Kikuchi approximation, the free energy is approximated as a sum of the free energies of basic clusters of nodes, minus the free energy of over-counted cluster intersections, minus the free energy of the over-counted intersections of intersections, and so on.

Let $R$ be a set of regions that include some chosen basic clusters of nodes, their intersections, the intersections of the intersections, and so on. The choice of basic clusters determines the Kikuchi approximation–for the Bethe approximation, the basic clusters consist of all linked pairs of nodes. Let $x_r$ be the state of the nodes in region $r$ and $b_r(x_r)$ be the "belief" in $x_r$. We define the energy of a region by

$E_r(x_r) \equiv -\ln \prod_{ij} \psi_{ij}(x_i, x_j) - \ln \prod_i \psi_i(x_i) \equiv -\ln \psi_r(x_r)$, where the products are over all interactions contained within the region $r$. For models with higher than pair-wise interactions, the region energy is generalized to include those interactions as well.

The Kikuchi free energy is

$$F_K = \sum_{r \in R} c_r \left( \sum_{x_r} b_r(x_r) E_r(x_r) + \sum_{x_r} b_r(x_r) \log b_r(x_r) \right) \qquad (6)$$

where $c_r$ is the over-counting number of region $r$, defined by: $c_r = 1 - \sum_{s \in super(r)} c_s$ where $super(r)$ is the set of all super-regions of $r$. For the largest regions in $R$, $c_r = 1$. The belief $b_r(\alpha_r)$ in region $r$ has several constraints: it must sum to one and be consistent with the beliefs in regions which intersect with $r$. In general, increasing the size of the basic clusters improves the approximation one obtains by minimizing the Kikuchi free energy.

## 4 Generalized belief propagation (GBP)

Minimizing the Kikuchi free energy subject to the constraints on the beliefs is not simple. Nearly all applications of the Kikuchi approximation in the physics literature exploit symmetries in the underlying physical system and the choice of clusters to reduce the number of equations that need to be solved from $O(N)$ to $O(1)$. But just as the Bethe free energy can be minimized by the BP algorithm, we introduce a class of analogous *generalized belief propagation* (GBP) algorithms that minimize an arbitrary Kikuchi free energy. These algorithms represent an advance in physics, in that they open the way to the exploitation of Kikuchi approximations for inhomogeneous physical systems.

There are in fact many possible GBP algorithms which all correspond to the same Kikuchi approximation. We present a "canonical" GBP algorithm which has the nice property of reducing to ordinary BP at the Bethe level. We introduce messages $m_{rs}(x_s)$ between all regions $r$ and their "direct sub-regions" $s$. (Define the set $sub_d(r)$ of direct sub-regions of $r$ to be those regions that are sub-regions of $r$ but have no super-regions that are also sub-regions of $r$, and similarly for the set $super_d(r)$ of "direct super-regions.") It is helpful to think of this as a message from those nodes in $r$ but not in $s$ (which we denote by $r \backslash s$) to the nodes in $s$. Intuitively, we want messages to propagate information that lies outside of a region into it. Thus, for a given region $r$, we want the belief $b_r(x_r)$ to depend on exactly those messages $m_{r's'}$ that start outside of the region $r$ and go into the region $r$. We define this set of messages $M(r)$ to be those messages $m_{r's'}(x_{s'})$ such that region $r' \backslash s'$ has no nodes in common with region $r$, and such that region $s'$ is a sub-region of $r$ or the same as region $r$. We also define the set $M(r, s)$ of messages to be all those messages that start in a sub-region of $r$ and also belong to $M(s)$, and we define $M(r) \backslash M(s)$ to be those messages that are in $M(r)$ but not in $M(s)$.

The canonical generalized belief propagation update rules are:

$$m_{rs} \leftarrow \alpha \left[ \sum_{x_{r \backslash s}} \psi_{r \backslash s}(x_{r \backslash s}) \prod_{m_{r''s''} \in M(r) \backslash M(s)} m_{r''s''} \right] / \prod_{m_{r's'} \in M(r,s)} m_{r's'} \qquad (7)$$

$$b_r \leftarrow \alpha \psi_r(x_r) \prod_{m_{r's'} \in M(r)} m_{r's'} \qquad (8)$$

where for brevity we have suppressed the functional dependences of the beliefs and messages. The messages are updated starting with the messages into the smallest

regions first. One can then use the newly computed messages in the product over $M(r, s)$ of the message-update rule. Empirically, this helps convergence.

*Claim 2:* Let $\{m_{rs}(x_s)\}$ be a set of canonical GBP messages and let $\{b_r(x_r)\}$ be the beliefs calculated from those messages. Then the beliefs are fixed-points of the canonical GBP algorithm if and only if they are zero gradient points of the constrained Kikuchi free energy $F_K$.

We prove this claim by adding Lagrange multipliers: $\gamma_r$ to enforce the normalization of $b_r$ and $\lambda_{rs}(x_s)$ to enforce the consistency of each region $r$ with all of its direct sub-regions $s$. This set of consistency constraints is actually more than sufficient, but there is no harm in adding extra constraints. We then rotate to another set of Lagrange multipliers $\mu_{rs}(x_s)$ of equal dimensionality which enforce a linear combination of the original constraints: $\mu_{rs}(x_s)$ enforces all those constraints involving marginalizations by all direct super-regions $r'$ of $s$ into $s$ except that of region $r$ itself. The rotation matrix is in a block form which can be guaranteed to be full rank. We can then show that the $\mu_{rs}(x_s)$ constraints can be written in the form $\mu_{rs}(x_s) \sum_{r' \in R(\mu_{rs})} c_{r'} \sum_{x_{r'}} b(x_{r'}')$ where $R(\mu_{rs})$ is the set of all regions which receive the message $\mu_{rs}$ in the belief update rule of the canonical algorithm. We then re-arrange the sum over all $\mu$'s into a sum over all regions, which has the form $\sum_{r \in R} c_r \sum_{x_r} b_r(x_r) \sum_{\mu_{rs} \in \tilde{M}(r)} \mu_{rs}(x_s)$. ($\tilde{M}(r)$ is a set of $\mu_{r's'}$ in one-to-one correspondence with the $m_{r's'}$ in $M(r)$.) Finally, we differentiate the Kikuchi free energy with respect to $b_r(r)$, and identify $\mu_{rs}(x_s) = \ln m_{rs}(x_s)$ to obtain the canonical GBP belief update rules, Eq. 8. Using the belief update rules in the marginalization constraints, we obtain the canonical GBP message update rules, Eq. 7.

It is clear from this proof outline that other GBP message passing algorithms which are equivalent to the Kikuchi approximation exist. If one writes any set of constraints which are sufficient to insure the consistency of all Kikuchi regions, one can associate the exponentiated Lagrange multipliers of those constraints with a set of messages.

The GBP algorithms we have described solve exactly those networks which have the topology of a tree of basic clusters. This is reminiscent of Pearl's method of clustering [8], wherein wherein one groups clusters of nodes into "super-nodes," and then applies a belief propagation method to the equivalent super-node lattice. We can show that the clustering method, using Kikuchi clusters as super-nodes, also gives results equivalent to the Kikuchi approximation for those lattices and cluster choices where there are no intersections between the intersections of the Kikuchi basic clusters. For those networks and cluster choices which do not obey this condition, (a simple example that we discuss below is the square lattice with clusters that consist of all square plaquettes of four nodes), Pearl's clustering method must be modified by adding additional update conditions to agree with GBP algorithms and the Kikuchi approximation.

# 5   Application to Specific Lattices

We illustrate the canonical GBP algorithm for the Kikuchi approximation of overlapping 4-node clusters on a square lattice of nodes. Figure 1 (a), (b), (c) illustrates the beliefs at a node, pair of nodes, and at a cluster of 4 nodes, in terms of messages propagated in the network. Vectors are the single index messages also used in ordinary BP. Vectors with line segments indicate the double-indexed messages arising from the Kikuchi approximation used here. These can be thought of as correction terms accounting for correlations between messages that ordinary BP treats as in-

dependent. (For comparison, Fig. 1 (d), (e), (f) shows the corresponding marginal computations for the triangular lattice with all triangles chosen as the basic Kikuchi clusters).

We find the message update rules by equating marginalizations of Fig. 1 (b) and (c) with the beliefs in Fig. 1 (a) and (b), respectively. Figure 2 (a) and (b) show (graphically) the resulting fixed point equations. The update rule (a) is like that for ordinary BP, with the addition of two double-indexed messages. The update rule for the double-indexed messages involves division by the newly-computed single-indexed messages. Fixed points of these message update equations give beliefs that are stationary points (empirically minima) of the corresponding Kikuchi approximation to the free energy.

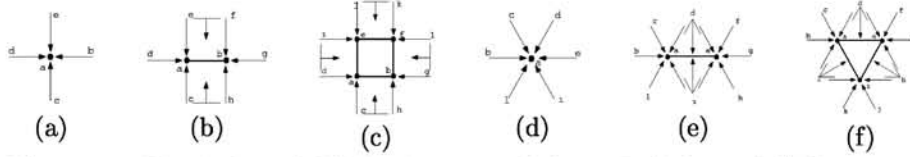

(a)        (b)        (c)        (d)        (e)        (f)

**Figure 1:** Marginal probabilities in terms of the node links and GBP messages. For (a) node, (b) line, (c) square cluster, using a Kikuchi approximation with 4-node clusters on a square lattice. E.g., (b) depicts (a special case of Eq. 8, written here using node labels): $b_{ab}(x_a, x_b) = \alpha\psi_{ab}(x_a, x_b)\psi_a(x_a)\psi_b(x_b)M_a^c M_a^d M_a^e M_{ab}^{ef} M_b^f M_b^g M_b^h M_{ab}^{ch}$, where super and subscripts indicate which nodes message $M$ goes from and to. (d), (e), (f): Marginal probabilities for triangular lattice with 3-node Kikuchi clusters.

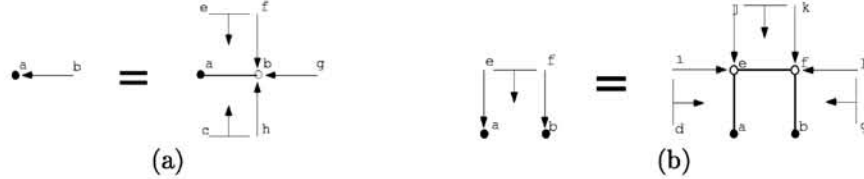

(a)                                        (b)

**Figure 2:** Graphical depiction of message update equations (Eq. 7; marginalize over nodes shown unfilled) for GBP using overlapping 4-node Kikuchi clusters. (a) Update equation for the single-index messages: $M_a^b(x_a) = \alpha\sum_{x_b}\psi_b(x_b)\psi_{ab}(x_a, x_b)M_{ab}^{ef} M_b^f M_b^g M_b^h M_{ab}^{ch}$. (b) Update equation for double-indexed messages (involves a division by the single-index messages on the left hand side).

## 6   Experimental Results

Ordinary BP is expected to perform relatively poorly for networks with many tight loops, conflicting interactions, and weak evidence. We constructed such a network, known in the physics literature as the square lattice Ising spin glass in a random magnetic field. The nodes are on a square lattice, with nearest neighbor nodes connected by a compatibility matrix of the form $\psi_{ij} = \begin{pmatrix} \exp(J_{ij}) & \exp(-J_{ij}) \\ \exp(-J_{ij}) & \exp(J_{ij}) \end{pmatrix}$ and local evidence vectors of the form $\psi_i = (\exp(h_i); \exp(-h_i))$. To instantiate a particular network, the $J_{ij}$ and $h_i$ parameters are chosen randomly and independently from zero-mean Gaussian probability distributions with standard deviations $J$ and $h$ respectively.

The following results are for $n$ by $n$ lattices with toroidal boundary conditions and with $J = 1$, and $h = 0.1$. This model is designed to show off the weaknesses of ordinary BP, which performs well for many other networks. Ordinary BP is a special case of canonical GBP, so we exploited this to use the same general-purpose GBP code for both ordinary BP and canonical GBP using overlapping square four-node clusters, thus making computational cost comparisons reasonable. We started with randomized messages and only stepped half-way towards the computed values of the messages at each iteration in order to help convergence. We found that canonical GBP took about twice as long as ordinary BP per iteration, but would typically reach a given level of convergence in many fewer iterations. In fact, for the majority of the dozens of samples that we looked at, BP did not converge at all, while canonical GBP always converged for this model and always to accurate answers. (We found that for the zero-field 3-dimensional spin glass with toroidal boundary conditions, which is an even more difficult model, canonical GBP with 2x2x2 cubic clusters would also fail to converge).

For $n = 20$ or larger, it was difficult to make comparisons with any other algorithm, because ordinary BP did not converge and Monte Carlo simulations suffered from extremely slow equilibration. However, generalized belief propagation converged reasonably rapidly to plausible-looking beliefs. For small $n$, we could compare with exact results, by using Pearl's clustering method on a chain of $n$ by 1 super-nodes. To give a qualitative feel for the results, we compare ordinary BP, canonical GBP, and the exact results for an $n = 10$ lattice where ordinary BP did converge. Listing the values of the one-node marginal probabilities in one of the rows, we find that ordinary BP gives (.0043807, .74502, .32866, .62190, .37745, .41243, .57842, .74555, .85315, .99632), canonical GBP gives (.40255, .54115, .49184, .54232, .44812, .48014, .51501, .57693, .57710, .59757), and the exact results were (.40131, .54038, .48923, .54506, .44537, .47856, .51686, .58108, .57791, .59881).

# References

[1] W. T. Freeman and E. Pasztor. Learning low-level vision. In *7th Intl. Conf. Computer Vision*, pages 1182–1189, 1999.

[2] B. J. Frey. *Graphical Models for Machine Learning and Digital Communication*. MIT Press, 1998.

[3] M. Jordan, Z. Ghahramani, T. Jaakkola, and L. Saul. An introduction to variational methods for graphical models. In M. Jordan, editor, *Learning in Graphical Models*. MIT Press, 1998.

[4] Y. Kabashima and D. Saad. Belief propagation vs. TAP for decoding corrupted messages. *Euro. Phys. Lett.*, 44:668, 1998.

[5] R. Kikuchi. *Phys. Rev.*, 81:988, 1951.

[6] R. McEliece, D. MacKay, and J. Cheng. Turbo decoding as an instance of Pearl's 'belief propagation' algorithm. *IEEE J. on Sel. Areas in Comm.*, 16(2):140–152, 1998.

[7] K. Murphy, Y. Weiss, and M. Jordan. Loopy belief propagation for approximate inference: an empirical study. In *Proc. Uncertainty in AI*, 1999.

[8] J. Pearl. *Probabilistic reasoning in intelligent systems: networks of plausible inference*. Morgan Kaufmann, 1988.

[9] T. J. Richardson. The geometry of turbo-decoding dynamics. *IEEE Trans. Info. Theory*, 46(1):9–23, Jan. 2000.

[10] Special issue on Kikuchi methods. Progr. Theor. Phys. Suppl., vol. 115, 1994.
